# Constant-Time Loading of Shallow 1-Dimensional Networks

**Stephen Judd**
Siemens Corporate Research,
755 College Rd. E.,
Princeton, NJ 08540
judd@learning.siemens.com

## Abstract

The complexity of learning in shallow 1-Dimensional neural networks has been shown elsewhere to be linear in the size of the network. However, when the network has a huge number of units (as cortex has) even linear time might be unacceptable. Furthermore, the algorithm that was given to achieve this time was based on a single serial processor and was biologically implausible.

In this work we consider the more natural parallel model of processing and demonstrate an expected-time complexity that is *constant* (i.e. independent of the size of the network). This holds even when inter-node communication channels are short and local, thus adhering to more biological and VLSI constraints.

## 1  Introduction

Shallow neural networks are defined in [Jud90]; the definition effectively limits the depth of networks while allowing the width to grow arbitrarily, and it is used as a model of neurological tissue like cortex where neurons are arranged in arrays tens of millions of neurons wide but only tens of neurons deep. Figure 1 exemplifies a family of networks which are not only shallow but "1-dimensional" as well—we allow the network to be extended as far as one liked in width (i.e. to the right) by repeating the design segments shown. The question we address is how learning time scales with the width. In [Jud88], it was proved that the worst case time complexity

of training this family is linear in the width. But the proof involved an algorithm that was biologically very implausible and it is this objection that will be somewhat redressed in this paper.

The problem with the given algorithm is that it operates only a monolithic serial computer; the single-CPU model of computing has no overt constraints on communication capacities and therefore is too liberal a model to be relevant to our neural machinery. Furthermore, the algorithm reveals very little about how to do the processing in a parallel and distributed fashion. In this paper we alter the model of computing to attain a degree of biological plausibility. We allow a linear number processors and put explicit constraints on the time required to communicate between processors. Both of these changes make the model much more biological (and also closer to the connectionist style of processing).

This change alone, however, does not alter the time complexity—the worst case training time is still linear. But when we change the complexity question being asked, a different answer is obtained. We define a class of tasks (viz. training data) that are drawn at random and then ask for the *expected* time to load these tasks, rather than the worst-case time. This alteration makes the question much more environmentally relevant. It also leads us into a different domain of algorithms and yields fast loading times.

## 2    Shallow 1-D Loading

### 2.1    Loading

A *family* of the example shallow 1-dimensional architectures that we shall examine is characterized solely by an integer, $d$, which defines the depth of each architecture in the family. An example is shown in figure 1 for $d = 3$. The example also happens to have a fixed fan-in of 2 and a very regular structure, but this is not essential. A *member* of the family is specified by giving the width $n$, which we will take to be the number of output nodes.

A *task* is a set of pairs of binary vectors, each specifying an stimulus to a net and its desired response. A random task of size $t$ is a set of $t$ pairs of independently drawn random strings; there is no guarantee it is a function.

Our primary question has to do with the following problem, which is parameterized by some fixed depth $d$, and by a *node function set* (which is the collection of different transfer functions that a node can be tuned to perform):

**Shallow 1-D Loading :**
Instance: An integer $n$, and a task.
Objective: Find a function (from the node function set) for each node in the network in the shallow 1-D architecture defined by $d$ and $n$ such that the resulting circuit maps all the stimuli in the task to their associated responses.

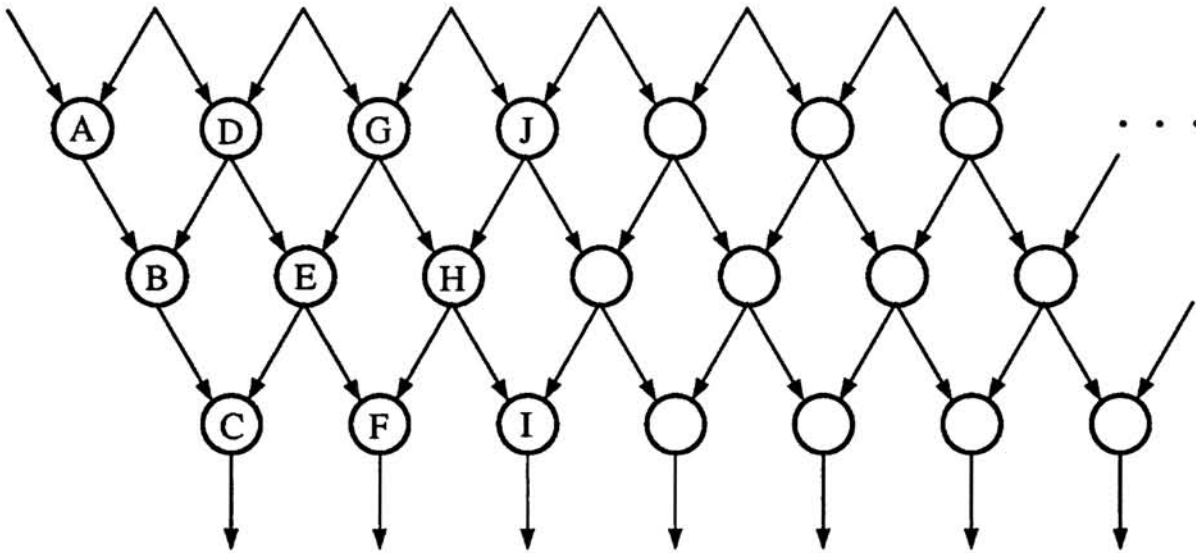

Figure 1: A Example Shallow 1-D Architecture

## 2.2   Model of Computation

Our machine model for solving this question is the following: For an instance of shallow 1-D loading of width $n$, we allow $n$ processors. Each one has access to a piece of the task, namely processor $i$ has access to bits $i$ through $i + d$ of each stimulus, and to bit $i$ of each response. Each processor $i$ has a communication link only to its two neighbours, namely processors $i - 1$ and $i + 1$. (The first and $n^{th}$ processors have only one neighbour.) It takes one time step to communicate a fixed amount of data between neighbours. There is no charge for computation, but this is not an unreasonable cheat because we can show that a matrix multiply is sufficient for this problem, and the size of the matrix is a function only of $d$ (which is fixed).

This definition accepts the usual connectionist ideal of having the processor closely identified with the network nodes for which it is "finding the weights", and data available at the processor is restricted to the same "local" data that connectionist machines have.

This sort of computation sets the stage for a complexity question,

## 2.3   Question and Approach

We wish to demonstrate that

**Claim 1** *This parallel machine solves shallow 1-D loading where each processor is finished in constant expected time The constant is dependent on the depth of the architecture and on the size of the task, but not on the width. The expectation is over the tasks.*

For simplicity we shall focus on one particular processor—the one at the leftmost end—and we shall further restrict our attention to finding a node function for one particular node.

To operate in parallel, it is necessary and sufficient for each processor to make its local decisions in a "safe" manner—that is, it must make choices for its nodes in such a way as to facilitate a global solution. Constant-time loading precludes being able to see all the data; and if only local data is accessible to a processor, then its plight is essentially to find an assignment that is compatible with *all* nonlocal satisfying assignments.

**Theorem 2** *The expected communication complexity of finding a "safe" node function assignment for a particular node in a shallow 1-D architecture is a constant dependent on d and t, but not on n.*

If decisions about assignments to single nodes can be made easily and essentially without having to communicate with most of the network, then the induced partitioning of the problem admits of fast parallel computation. There are some complications to the details because all these decisions must be made in a coordinated fashion, but we omit these details here and claim they are secondary issues that do not affect the gross complexity measurements.

The proof of the theorem comes in two pieces. First, we define a computational problem called path finding and the graph-theoretic notion of domination which is its fundamental core. Then we argue that the loading problem can be reduced to path finding in constant parallel time and give an upper bound for determining domination.

# 3   Path Finding

The following problem is parameterized by an integer $K$, which is fixed.

**Path finding :**
    Instance: An integer $n$ defining the number of parts in a partite graph, and a series of $K \times K$ adjacency matrices, $M_1, M_2, \ldots M_{n-1}$. $M_i$ indicates connections between the K nodes of part $i$ and the $K$ nodes of part $i+1$.
    Objective: Find a path of $n$ nodes, one from each part of the $n$-partite graph.

Define $X_h$ to be the binary matrix representing connectivity between the first part of the graph and the $i^{\text{th}}$ part: $X_1 = M_1$ and $X_h(j, k) = 1$ iff $\exists m$ such that $X_h(j, m) = 1$ and $M_h(m, k) = 1$. We say "*i includes j at h*" if every bit in the $i^{\text{th}}$ row of $X_h$ is 1 whenever the corresponding bit in the $j^{\text{th}}$ row of $X_h$ is 1. We say "*i dominates at h*" or "*i is a dominator*" if for all rows $j$, $i$ includes $j$ at $h$.

**Lemma 3** *Before an algorithm can select a node i from the first part of the graph to be on the path, it is necessary and sufficient for i to have been proven to be a dominator at some h.*                                                                □

The minimum $h$ required to prove domination stands as our measure of "communication complexity".

**Lemma 4** *Shallow 1-D Loading can be reduced to path finding in constant parallel time.*

**Proof**: Each output node in a shallow architecture has a set of nodes leading into it called a support cone (or "receptive field"), and the collection of functions assigned to those nodes will determine whether or not the output bit is correct in each response. Nodes A,B,C,D,E,G in Figure 1 are the support cone for the first output node (node C), and D,E,F,G,H,J are the cone for the second. Construct each part of the graph as a set of points each corresponding to an assignment over the whole support cone that makes its output bit always correct. This can be done for each cone in parallel, and since the depth (and the fan-in) is fixed, the set of all possible assignments for the support cone can be enumerated in constant time. Now insert edges between adjacent parts wherever two points correspond to assignments that are mutually compatible. (Note that since the support cones overlap one another, we need to ensure that assignments are consistent with each other.) This also can be done in constant parallel time. We call this construction a *compatibility graph*.

A solution to the loading problem corresponds exactly to a path in the compatibility graph. □

A dominator in this path-finding graph is exactly what was meant above by a "safe" assignment in the loading problem.

## 4  Proof of Theorem

Since it is possible that there is no assignments to certain cones that correctly map the stimuli it is trivial to prove the theorem, but as a practical matter we are interested in the case where the architecture is actually capable of performing the task. We will prove the theorem using a somewhat more satisfying event.

**Proof** of theorem 2: For each support cone there is 1 output bit per response and there are $t$ such responses. Given the way they are generated, these responses could all be the same with probability $.5^{t-1}$. The probability of two adjacent cones both having to perform such a constant mapping is $.5^{2(t-1)}$.

Imagine the labelling in Figure 1 to be such that there were many support cones to the left (and right) of the piece shown. Any path through the left side of the compatibility graph that arrived at some point in the part for the cone to the left of C would imply an assignment for nodes A, B, and D. Any path through the right side of the compatibility graph that arrived at some point in the part for the cone of I would imply an assignment for nodes G, H, and J. If cones C and F were both required to merely perform constant mappings, then any and all assignments to A, B, and D would be compatible with any and all assignments to G, H, and J (because nodes C and F could be assigned constant functions themselves, thereby making the others irrelevant). This insures that any point on a path to the left will dominate at the part for I.

Thus $2^{2(t-1)}$ (the inverse of the probability of this happening) is an upper bound on the domination distance, i.e. the communication complexity, i.e. the loading time. □

More accurately, the complexity is $\min(c(d,t), f(t), n)$, where $c$ and $f$ are some unknown functions. But the operative term here is usually $c$ because $d$ is unlikely to get so large as to bring $f$ into play (and of course $n$ is unbounded).

The analysis in the proof is sufficient, but it is a far cry from complete. The actual Markovian process in the sequence of $X$'s is much richer; there are so many events in the compatibility graph that cause domination to occur that is takes a lot of careful effort to construct a task that will *avoid* it!

## 5  Measuring the Constants

Unfortunately, the very complications that give rise to the pleasant robustness of the domination event also make it fiendishly difficult to analyze quantitatively. So to get estimates for the actual constants involved we ran Monte Carlo experiments.

We ran experiments for 4 different cases. The first experiment was to measure the distance one would have to explore before finding a dominating assignment for the node labeled A in figure 1. The node function set used was the set of linearly separable functions. In all experiments, if domination occurred for the degenerate reason that there were no solutions (paths) at all, then that datum was thrown out and the run was restarted with a different seed.

Figure 2 reports the constants for the four cases. There is one curve for each experiment. The abscissa represents $t$, the size of the task. The ordinate is the number of support cones that must be consulted before domination can be expected to occur. All points given are the average of at least 500 trials. Since $t$ is an integer the data should not have been interpolated between points, but they are easier to see as connected lines. The solid line (labeled LSA) is for the case just described. It has a bell shape, reflecting three facts:

- when the task is very small almost every choice of node function for one node is compatible with choices for the neighbouring nodes.
- when the task is very large, there so many constraints on what a node must compute that it is easy to resolve what that should be without going far afield.
- when the task is intermediate-sized, the problem is harder.

Note the very low distances involved—even the peak of the curve is well below 2, so nowhere would you expect to have to pass data more than 2 support cones away. Although this worst-expected-case would surely be larger for deeper nets, current work is attempting to see how badly this would scale with depth (larger $d$).

The curve labeled LUA is for the case where all Boolean functions are used as the node function set. Note that it is significantly higher in the region $6 \leq t \leq 12$. The implication is that although the node function set being used here is a superset of the linearly separable functions, it takes *more* computation at loading time to be able to exploit that extra power.

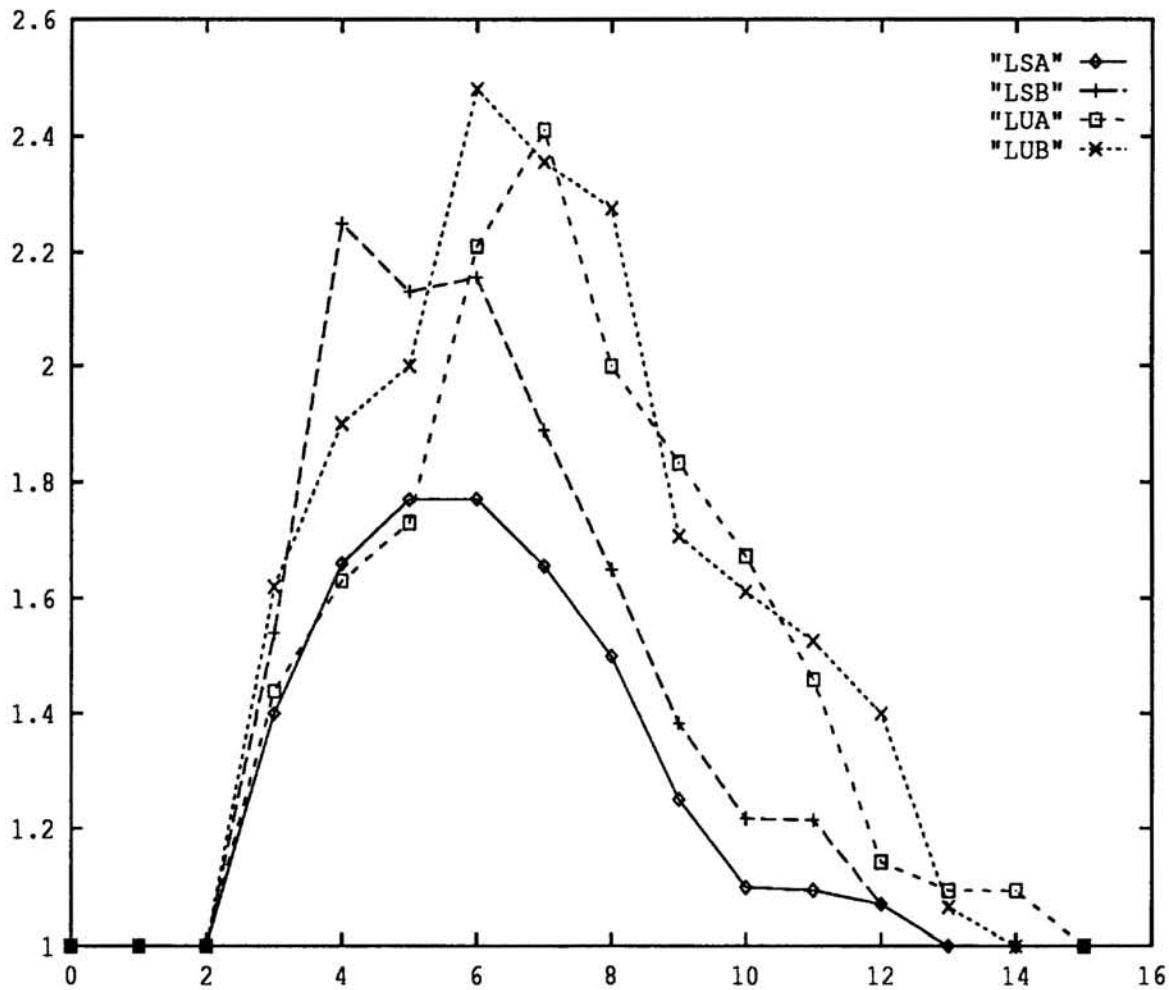

Figure 2: Measured Domination Distances.

The curve labeled LSB shows the expected distance one has to explore before finding a dominating assignment for the node labelled B in figure 1. The node function set used was the set of linearly separable functions. Note that it is everywhere higher that the LSA curve, indicating that the difficulty of settling on a correct node function for a second-layer node is somewhat higher than finding one for a first-layer node.

Finally, there is a curve for node B when all Boolean functions are used (LUB). It is generally higher than when just linearly separable functions are used, but not so markedly so as in the case of node A.

## 6    Conclusions

The model of computation used here is much more biologically relevant than the ones previously used for complexity results, but the algorithm used here runs in an off-line "batch mode" (i.e. it has all the data before it starts processing). This has an unbiological nature, but no more so than the customary connectionist habit of repeating the data many times.

A weakness of our analysis is that (as formulated here) it is only for discrete node functions, exact answers, and noise-free data. Extensions for any of these additional difficulties may be possible, and the bell shape of the curves should survive.

The peculiarities of the regular 3-layer network examined here may appear restrictive, but it was taken as an example only; what is really implied by the term "1-D" is only that the bandwidth of the SCI graph for the architecture be bounded (see [Jud90] for definitions). This constraint allows several degrees of freedom in choosing the architecture, but domination is such a robust combinatoric event that the essential observation about bell-shaped curves made in this paper will persist even in the face of large changes from these examples.

We suggest that whatever architectures and node function sets a designer cares to use, the notion of domination distance will help reveal important computational characteristics of the design.

## Acknowledgements

Thanks go to Siemens and CalTech for wads of computer time.

## References

[Jud88] J. S. Judd. On the complexity of loading shallow neural networks. *Journal of Complexity*, September 1988. Special issue on Neural Computation, in press.

[Jud90] J. Stephen Judd. *Neural Network Design and the Complexity of Learning.* MIT Press, Cambridge, Massachusetts, 1990.